# AER Building Blocks for Multi-Layer Multi-Chip Neuromorphic Vision Systems

**R. Serrano-Gotarredona[1], M. Oster[2], P. Lichtsteiner[2], A. Linares-Barranco[4], R. Paz-Vicente[4], F. Gómez-Rodríguez[4], H. Kolle Riis[3], T. Delbrück[2], S. C. Liu[2], S. Zahnd[2], A. M. Whatley[2], R. Douglas[2], P. Häfliger[3], G. Jimenez-Moreno[4], A. Civit[4], T. Serrano-Gotarredona[1], A. Acosta-Jiménez[1], B. Linares-Barranco[1]**

[1]Instituto de Microelectrónica de Sevilla (IMSE-CNM-CSIC) Sevilla Spain, [2]Institute of Neuroinformatics (INI-ETHZ) Zurich Switzerland, [3]University of Oslo Norway (UIO), [4]University of Sevilla Spain (USE).

## Abstract

A 5-layer neuromorphic vision processor whose components communicate spike events asynchronously using the address-event-representation (AER) is demonstrated. The system includes a retina chip, two convolution chips, a 2D winner-take-all chip, a delay line chip, a learning classifier chip, and a set of PCBs for computer interfacing and address space remappings. The components use a mixture of analog and digital computation and will learn to classify trajectories of a moving object. A complete experimental setup and measurements results are shown.

## 1 Introduction

The Address-Event-Representation (AER) is an event-driven asynchronous inter-chip communication technology for neuromorphic systems [1][2]. Senders (e.g. pixels or neurons) asynchronously generate events that are represented on the AER bus by the source addresses. AER systems can be easily expanded. The events can be merged with events from other senders and broadcast to multiple receivers [3]. Arbitrary connections, remappings and transformations can be easily performed on these digital addresses.

A potentially huge advantage of AER systems is that computation is event driven and thus can be very fast and efficient. Here we describe a set of AER building blocks and how we assembled them into a prototype vision system that learns to classify trajectories of a moving object. All modules communicate asynchronously using AER. The building blocks and demonstration system have been developed in the EU funded research project CAVIAR (Convolution AER VIsion Architecture for Real-time). The building blocks (Fig. 1) consist of: (1) a retina loosely modeled on the magnocellular pathway that responds to brightness changes, (2) a convolution chip with programmable convolution kernel of arbitrary shape and size, (3) a multi-neuron 2D competition chip, (4) a spatio-temporal pattern classification learning module, and (5) a set of FPGA-based PCBs for address remapping and computer interfaces.

Using these AER building blocks and tools we built the demonstration vision system shown schematically in Fig. 1, that detects a moving object and learns to classify its

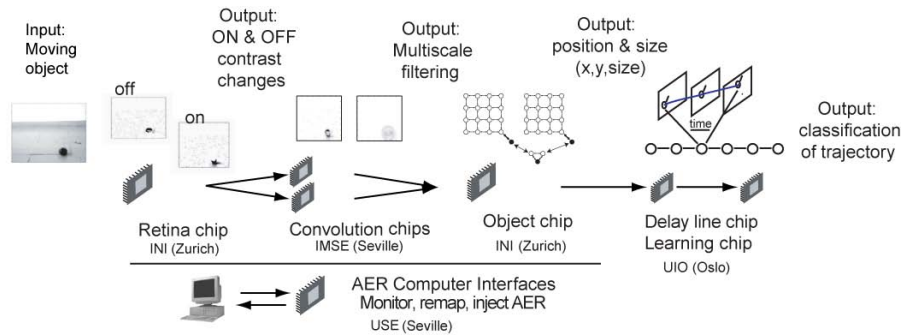

**Fig. 1: Demonstration AER vision system**

trajectories. It has a front end retina, followed by an array of convolution chips, each programmed to detect a specific feature with a given spatial scale. The competition or 'object' chip selects the most salient feature and scale. A spatio-temporal pattern classification module categorizes trajectories of the object chip outputs.

## 2  Retina

Biological vision uses asynchronous events (spikes) delivered from the retina. The stream of events encodes dynamic scene contrast. Retinas are optimized to deliver relevant information and to discard redundancy. CAVIAR's input is a dynamic visual scene. We developed an AER silicon retina chip 'TMPDIFF' that generates events corresponding to *relative changes in image intensity* [8]. These address-events are broadcast asynchronously on a shared digital bus to the convolution chips. Static scenes produce no output. The events generated by TMPDIFF represent relative changes in intensity that exceed a user-defined threshold and are ON or OFF type depending on the sign of the change since the last event. This silicon retina loosely models the magnocellular retinal pathway.

The front-end of the pixel core (see Fig. 2a) is an active unity-gain logarithmic photoreceptor that can be self-biased by the average photocurrent [7]. The active feedback speeds up the response compared to a passive log photoreceptor and greatly increases bandwidth at low illumination. The photoreceptor output is buffered to a voltage-mode capacitive-feedback amplifier with closed-loop gain set by a well-matched capacitor ratio. The amplifier is balanced after transmission of each event by the AER handshake. ON and OFF events are detected by the comparators that follow. Mismatch of the event threshold is determined by only 5 transistors and is effectively further reduced by the gain of the amplifier. Much higher contrast resolution than in previous work [6] is obtained by using the excellent matching between capacitors to form a self-clocked switched-capacitor change amplifier, allowing for operation with scene contrast down to about 20%. A chip photo is shown in Fig. 2b.

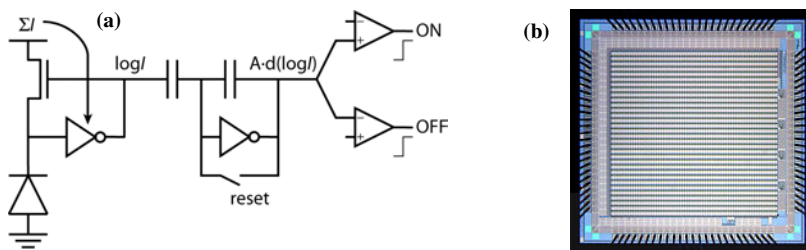

**Fig. 2.  Retina. a) core of pixel circuit, b) chip photograph.**

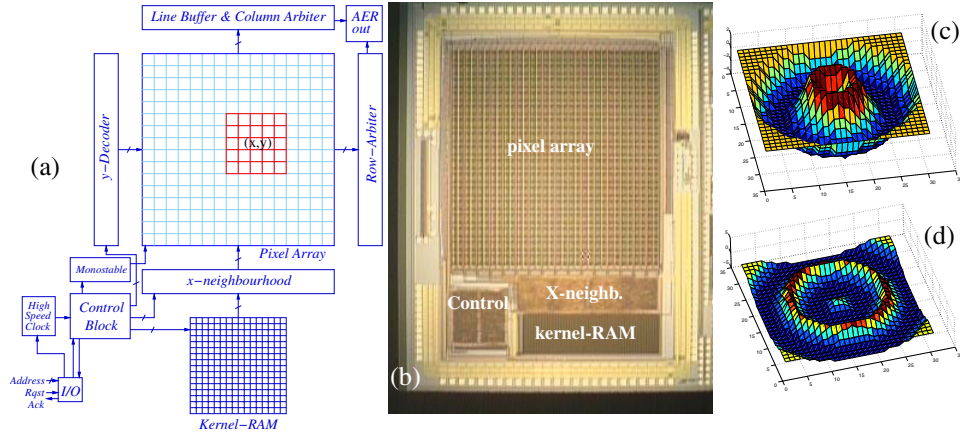

**Fig. 3.** Convolution chip (a) architecture of the convolution chip. (b) microphotograph of fabricated chip. (c) kernel for detecting circumferences of radius close to 4 pixels and (d) close to 9 pixels.

TMPDIFF has 64x64 pixels, each with 2 outputs (ON and OFF), which are communicated off-chip on a 16-bit AER bus. It is fabricated in a 0.35μm process. Each pixel is 40x40 μm$^2$ and has 28 transistors and 3 capacitors. The operating range is at least 5 decades and minimum scene illumination with $f$/1.4 lens is less than 10 lux.

## 3   Convolution Chip

The convolution chip is an AER transceiver with an array of event integrators. Foreach incoming event, integrators within a projection field around the addressed pixel compute a weighted event integration. The weight of this integration is defined by the convolution kernel [4]. This event-driven computation puts the kernel onto the integrators.

Fig. 3a shows the block diagram of the convolution chip. The main parts of the chip are: (1) An array of 32x32 pixels. Each pixel contains a binary weighted signed current source and an integrate-and-fire signed integrator [5]. The current source is controlled by the kernel weight read from the RAM and stored in a dynamic register. (2) A 32x32 kernel RAM. Each kernel weight value is stored with signed 4-bit resolution. (3) A digital controller handles all sequence of operations. (4) A monostable. For each incoming event, it generates a pulse of fixed duration that enables the integration simultaneously in all the pixels. (5) X-Neighborhood Block. This block performs a displacement of the kernel in the x direction. (6) Arbitration and decoding circuitry that generate the output address events. It uses Boahen's burst mode fully parallel AER [2].

The chip operation sequence is as follows: (1) Each time an input address event is received, the digital control block stores the (x,y) address and acknowledges reception of the event. (2) The control block computes the x-displacement that has to be applied to the kernel and the limits in the y addresses where the kernel has to be copied. (3) The Afterwards, the control block generates signals that control on a row-by-row basis the copy of the kernel to the corresponding rows in the pixel array. (4) Once the kernel copy is finished, the control block activates the generation of a monostable pulse. This way, in each pixel a current weighted by the corresponding kernel weight is integrated during a fixed time interval. Afterwards, kernel weights in the pixels are erased. (5) When the integrator voltage in a pixel reaches a threshold, that pixel asynchronously sends an event, which is arbitrated and decoded in the periphery of the array. The pixel voltage is reset upon reception of the acknowledge from the periphery.

A prototype convolution chip has been fabricated in a CMOS 0.35μm process. Both the size of the pixel array and the size of the kernel storage RAM are 32x32. The input address space can be up to 128x128. In the experimental setup of Section 7, the 64x64 retina output is fed to the convolution chip, whose pixel array addresses are centered on that of the retina. The pixel size is 92.5μm x 95μm. The total chip area is 5.4x4.2 mm$^2$. Fig. 3b shows the microphotograph of the fabricated chip. AER events can be fed-in up to a peak rate of 50 Mevent/s. Output event rate depends on kernel lines $n_k$. The measured output AER peak delay is $(40 + 20 \times n_k)$ ns/event.

## 4 Competition 'Object' Chip

This AER transceiver chip consists of a group of VLSI integrate-and-fire neurons with various types of synapses [9]. It reduces the dimensionality of the input space by preserving the strongest input and suppressing all other inputs. The strongest input is determined by configuring the architecture on the 'Object' chip as a spiking winner-take-all network. Each convolution chip convolves the output spikes of the retina with its preprogrammed feature kernel (in our example, this kernel consists of a ring filter of a particular resolution). The 'Object' chip receives the outputs of several convolution chips and computes the winner (strongest input) in two dimensions. First, it determines the strongest input in each feature map and in addition, it determines the strongest feature. The computation to determine the strongest input in each feature map is carried out using a two-dimensional winner-take-all circuit as shown in Fig. 4. The network is configured so that it implements a hard winner-take-all, that is, only one neuron is active at a time. The activity of the winner is proportional to the winner's input activity.

The winner-take-all circuit can reliably select the winner given a difference of input firing rate of only 10% assuming that it receives input spike trains having a regular firing rate [10]. Each excitatory input spike charges the membrane of the post-synaptic neuron until one neuron in the array--the winner--reaches threshold and is reset. All other neurons are then inhibited via a global inhibitory neuron which is driven by all the excitatory neurons. Self-excitation provides hysteresis for the winning neuron by facilitating the selection of this neuron as the next winner.

Because of the moving stimulus, the network has to determine the winner using an estimate of the instantaneous input firing rates. The number of spikes that the neuron must integrate before eliciting an output spike can be adjusted by varying the efficacies of the input synapses.

To determine the winning feature, we use the activity of the global inhibitory neuron (which reflects the activity of the strongest input within a feature map) of each feature map in a second layer of competition. By adding a second global inhibitory neuron to each feature map and by driving this neuron through the outputs of the first global inhibitory neurons of all feature maps, only the strongest feature map will survive. The output of the object chip will be spikes encoding the spatial location of the stimulus and the identity of the winning feature. (In the characterization shown in Section 7, the global competition was disabled, so both objects could be simultaneously localized by the object chip).

We integrated the winner-take-all circuits for four feature maps on a single chip with a total of 16x16 neurons; each feature uses an 8x8 array. The chip was fabricated in a 0.35 μm CMOS process with an area of 8.5 mm$^2$.

## 5 Learning Spatio-Temporal Pattern Classification

The last step of data reduction in the CAVIAR demonstrator is a subsystem that learns to classify the spatio-temporal patterns provided by the object chip. It consists of three

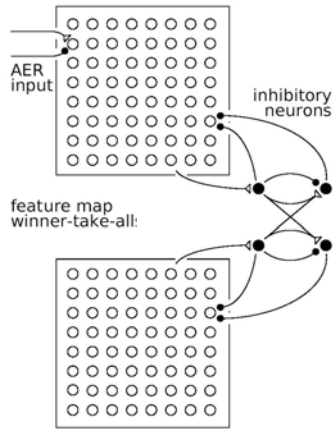

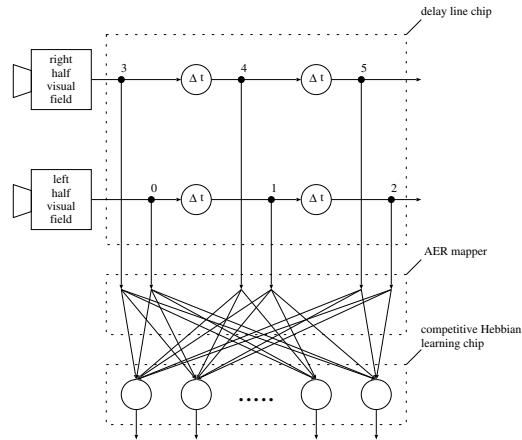

**Fig. 4: Architecture of 'Object' chip configured for competition within two feature maps and competition across the two feature maps.**

**Fig. 5: System setup for learning direction of motion**

components: a delay line chip, a competitive Hebbian learning chip [11], and an AER mapper that connects the two. The task of the delay line chip is to project the temporal dimension into a spatial dimension. The competitive Hebbian learning chip will then learn to classify the resulting patterns. The delay line chip consists of one cascade of 880 delay elements. 16 monostables in series form one delay element. The output of every delay element produces an output address event. A pulse can be inserted at every delay-element by an input address event. The cascade can be programmed to be interrupted or connected between any two subsequent delay-elements. The associative Hebbian learning chip consists of 32 neurons with 64 learning synapses each. Each synapse includes learning circuitry with a weak multi-level memory cell for spike-based learning [11].

A simple example of how this system may be configured is depicted in Fig. 5: the mapper between the object chip and the delay line chip is programmed to project all activity from the left half of the field of vision onto the input of one delay line, and from the right half of vision onto another. The mapper between the delay line chip and the competitive Hebbian learning chip taps these two delay lines at three different delays and maps these 6 outputs onto 6 synapses of each of the 32 neurons in the competitive Hebbian learning chip. This configuration lets the system learn the direction of motion.

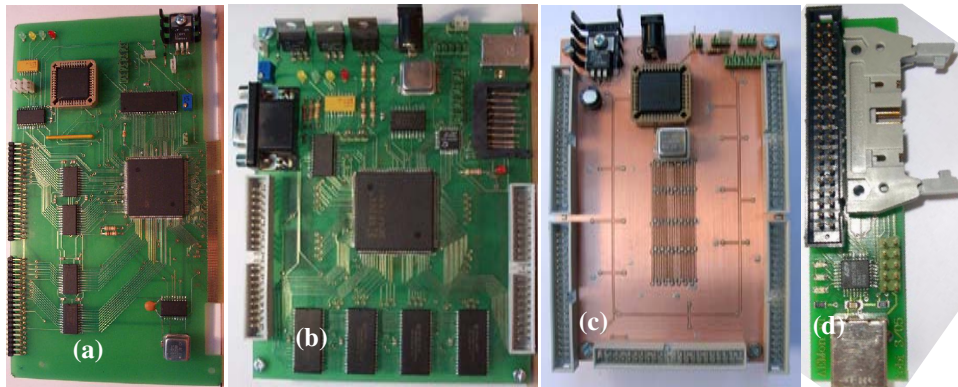

**Fig. 6: Developed AER interfacing PCBs. (a) PCI-AER, (b) USB-AER, (c) AER-switch, (d) mini-USB**

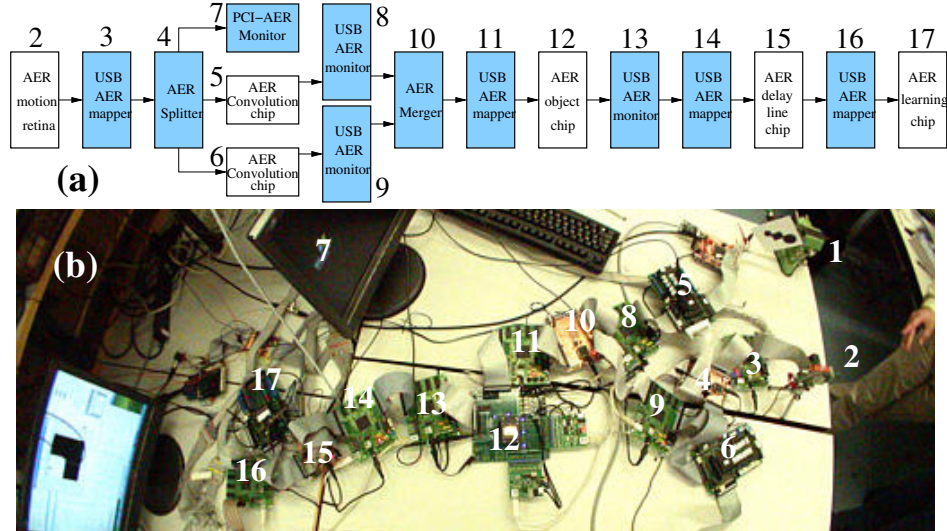

**Fig. 7: Experimental setup of multi-layered AER vision system for ball tracking (white boxes include custom designed chips, blue boxes are interfacing PCBs). (a) block diagram, (b) photograph of setup.**

## 6 Computer Interfaces

When developing and tuning complex hierarchical multi-chips AER systems it is crucial to have available proper computer interfaces for (a) reading AER traffic and visualizing it, and (b) for injecting synthesized or recorded AER traffic into AER buses. We developed several solutions. Fig. 6(a) shows a PCI-AER interfacing PCB capable of transmitting AER streams from within the computer or, vice versa, capturing them from an AER bus and into computer memory. It uses a Spartan-II FPGA, and can achieve a peak rate of 15 Mevent/s using PCI mastering. Fig. 5(b) shows a USB-AER board that does not require a PCI slot and can be controlled through a USB port. It uses a Spartan II 200 FPGA with a Silicon Labs C8051F320 microcontroller. Depending on the FPGA firmware, it can be used to perform five different functions: (a) transform sequence of frames into AER in real time [13], (b) histogram AER events into sequences of frames in real time, (c) do remappings of addresses based on look-up-tables, (d) capture timestamped events for off-line analysis, (e) reproduce time-stamped sequences of events in real time. This board can also work without a USB connection (stand-alone mode) by loading the firmware through MMC/SD cards, used in commercial digital cameras. This PCB can handle AER traffic of up to 25 Mevent/s. It also includes a VGA output for visualizing histogrammed frames. The third PCB, based on a simple CPLD, is shown in Fig. 6(c). It splits one AER bus into 2, 3 or 4 buses, and vice versa, merges 2, 3 or 4 buses into a single bus, with proper handling of handshaking signals. The last board in Fig. 6(d) is a lower performance but more compact single-chip bus-powered USB interface based on a C8051F320 microcontroller. It captures timestamped events to a computer at rates of up to 100 kevent/s and is particularly useful for demonstrations and field capture of retina output.

## 7 Demonstration Vision System

To test CAVIAR's capabilities, we built a demonstration system that could simultaneously track two objects of different size. A block diagram of the complete system is shown in Fig. 7(a), and a photograph of the complete experimental setup is given in Fig. 7(b). The

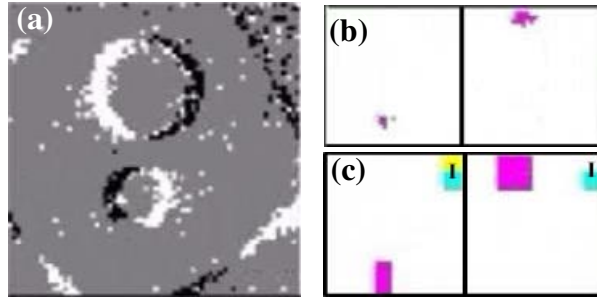

Fig. 8: Captured AER outputs at different stages of processing chain. (a) at the retina output, (b) at the output of the 2 convolution chips, (c) at the output of the object chip. 'I' labels the activity of the inhibitory neurons.

complete chain consisted of 17 pieces (chips and PCBs), all numbered in Fig. 7: (1) The rotating wheel stimulus. (2) The retina. The retina looked at a rotating disc with two solid circles on it of two different radii. (3) A USB-AER board as mapper to reassign addresses and eliminate the polarity of brightness change. (4) A 1-to-3 splitter (one output for the PCI-AER board (7) to visualize the retina output, as shown in Fig. 8(a), and two outputs for two convolution chips). (5-6) Two convolution chips programmed with the kernels in Fig. 3c-d, to detect circumferences of radius 4 pixels and 9 pixels, respectively. They see the complete 64x64 retina image (with rectified activity; polarity is ignored) but provide a 32x32 output for only the central part of the retina image. This eliminates convolution edge effects. The output of each convolution chip is fed to a USB-AER board working as a monitor (8-9) to visualize their outputs (Fig. 8b). The left half is for the 4-radius kernel and the right half for the 9-radius kernel. The outputs of the convolution chips provide the center of the circumferences only if they have radius close to 4 pixels or 9 pixels, respectively. As can be seen, each convolution chip detects correctly the center of its corresponding circumference, but not the other. Both chips are tuned for the same feature but with different spatial scale. Both convolution chips outputs are merged onto a single AER bus using a merger (10) and then fed to a mapper (11) to properly reassign the address and bit signs for the winner-take-all 'object' chip (12), which correctly decides the centers of the convolution chip outputs. The object chip output is fed to a monitor (13) for visualization purposes. This output is shown in Fig. 8(c). The output of this chip is transformed using a mapper (14) and fed to the delay line chip (15), the outputs of which are fed through a mapper (16) to the learning (17) chip. The system as characterized can simultaneously trach two objects of different shape; we have connected but not yet studied trajectory learning and classification.

## 8 Conclusions

In terms of the number of independent components, CAVIAR demonstrates the largest AER system yet assembled. It consists of 5 custom neuromorphic AER chips and at least 6 custom AER digital boards. Its functioning shows that AER can be used for assembling complex real time sensory processing systems and that relevant information about object size and location can be extracted and restored through a chain of feedforward stages. The CAVIAR system is a useful environment to develop reusable AER infrastructure and is capable of fast visual computation that is not limited by normal imager frame rate. Its continued development will result in insights about spike coding and representation.

## Acknowledgements

This work was sponsored by EU grant IST-2001-34124 (CAVIAR), and Spanish grant TIC-2003-08164-C03 (SAMANTA). We thank K. Boahen for sharing AER interface

technology and the EU project ALAVLSI for sharing chip development and other AER computer interfaces [14].

## References

[1] M. Sivilotti, *Wiring Considerations in Analog VLSI Systems with Application to Field-Programmable Networks*, Ph.D. Thesis, California Institute of Technology, Pasadena CA, 1991.

[2] K. Boahen, "Point-to-Point Connectivity Between Neuromorphic Chips Using Address Events," *IEEE Trans. on Circuits and Systems Part-II*, vol. 47, No. 5, pp. 416-434, May 2000.

[3] J. P. Lazzaro and J. Wawrzynek, "A Multi-Sender Asynchronous Extension to the Address-Event Protocol," *16th Conference on Advanced Research in VLSI*, W. J. Dally, J. W. Poulton, and A. T. Ishii (Eds.), pp. 158-169, 1995.

[4] T. Serrano-Gotarredona, A. G. Andreou, and B. Linares-Barranco, "AER Image Filtering Architecture for Vision Processing Systems," *IEEE Trans. Circuits and Systems (Part II): Analog and Digital Signal Processing*, vol. 46, No. 9, pp. 1064-1071, September 1999.

[5] R. Serrano-Gotarredona, B. Linares-Barranco, and T. Serrano-Gotarredona, "A New Charge-Packet Driven Mismatch-Calibrated Integrate-and-Fire Neuron for Processing Positive and Negative Signals in AER-based Systems," In *Proc. of the IEEE Int. Symp. Circ. Syst.,* (ISCAS04), vol. 5, pp. 744-747 ,Vancouver, Canada, May 2004.

[6] P. Lichtsteiner, T. Delbrück, and J. Kramer, "Improved ON/OFF temporally differentiating address-event imager," in *11th IEEE International Conference on Electronics, Circuits and Systems* (ICECS2004), Tel Aviv, Israel, 2004, pp. 211-214.

[7] T. Delbrück and D. Oberhoff, "Self-biasing low-power adaptive photoreceptor," in *Proc. of the IEEE Int. Symp. Circ. Syst.* (ISCAS04), pp. IV-844-847, 2004.

[8] P. Lichtsteiner and T. Delbrück "64x64 AER Logarithmic Temporal Derivative Silicon Retina," *Research in Microelectronics and Electronics*, Vol. 2, pp. 202-205, July 2005.

[9] Liu, S.-C. and Kramer, J. and Indiveri, G. and Delbrück, T. and Burg, T. and Douglas, R. "Orientation-selective aVLSI spiking neurons", *Neural Network*s, 14:(6/7) 629-643, Jul, 2001

[10] Oster, M. and Liu, S.-C. "A Winner-take-all Spiking Network with Spiking Inputs", in *11th IEEE International Conference on Electronics, Circuits and Systems* (ICECS 2004), Tel Aviv, pp. 203-206, 2004

[11] H. Kolle Riis and P. Haefliger, "Spike based learning with weak multi-level static memory," In *Proc. of the IEEE Int. Symp. Circ. Syst.* (ISCAS04), vol. 5, pp. 393-395, Vancouver, Canada, May 2004.

[12] P. Häfliger and H. Kolle Riis, "A Multi-Level Static Memory Cell," In *Proc. of the IEEE Int. Symp. Circ. Syst.* (ISCAS04), vol. 1, pp. 22-25, Bangkok, Thailand, May 2003.

[13] A. Linares-Barranco, G. Jiménez-Moreno, B. Linares-Barranco, and A. Civit-Ballcels, "On Algorithmic Rate-Coded AER Generation," accepted for publication in *IEEE Trans. Neural Networks*, May 2006 (tentatively).

[14] V. Dante, P. Del Giudice, and A. M. Whatley, "PCI-AER Hardware and Software for Interfacing to Address-Event Based Neuromorphic Systems", *The Neuromorphic Engineer*, 2:(1) 5-6, 2005.
